# Recurrent Cortical Amplification Produces Complex Cell Responses

**Frances S. Chance, Sacha B. Nelson, and L. F. Abbott**
Volen Center and Department of Biology
Brandeis University
Waltham, MA 02454

## Abstract

Cortical amplification has been proposed as a mechanism for enhancing the selectivity of neurons in the primary visual cortex. Less appreciated is the fact that the same form of amplification can also be used to de-tune or broaden selectivity. Using a network model with recurrent cortical circuitry, we propose that the spatial phase invariance of complex cell responses arises through recurrent amplification of feedforward input. Neurons in the network respond like simple cells at low gain and complex cells at high gain. Similar recurrent mechanisms may play a role in generating invariant representations of feedforward input elsewhere in the visual processing pathway.

## 1   INTRODUCTION

Synaptic input to neurons in the primary visual cortex is primarily recurrent, arising from other cortical cells. The dominance of this type of connection suggests that it may play an important role in cortical information processing. Previous studies proposed that recurrent connections amplify weak feedforward input to the cortex (Douglas et al., 1995) and selectively amplify tuning for specific stimulus characteristics, such as orientation or direction of movement (Douglas et al., 1995; Ben-Yishai et al., 1995; Somers et al., 1995; Sompolinsky and Shapley, 1997). Cortical cooling and shocking experiments provide evidence that there is cortical amplification through recurrent connections, but they do not show increases in orientation or direction selectivity as a result of this amplification (Ferster et al., 1996; Chung and Ferster, 1998). Recurrent connections can also decrease neuronal selectivity through the same form of amplification, generating responses that are insensitive to certain stimulus features. Although the ability to sharpen tuning may be an important feature in cortical processing, the capacity to broaden tuning for particular stimulus attributes is also desirable.

Neurons in the primary visual cortex can be divided into two classes based on their re-

sponses to visual stimuli such as counterphase and drifting sinusoidal gratings. Simple cells show tuning for orientation, spatial frequency, and spatial phase of a grating (Movshon et al., 1978a). Complex cells exhibit orientation and spatial frequency tuning, but are insensitive to spatial phase (Movshon et al., 1978b). A counterphase grating, $s(x,t) = \cos(Kx - \Phi)\cos(\omega t)$, is one in which the spatial phase, $\Phi$, and spatial frequency, $K$, are held constant but the contrast, $s(x,t)$, varies sinusoidally in time at some frequency $\omega$. In response to a counterphase grating, the activity of a simple cell oscillates at the same frequency as the stimulus, $\omega$. A complex cell response is modulated at twice the frequency, $2\omega$. To create a drifting grating of frequency $\nu$, $s(x,t) = \cos(Kx - \nu t)$, the spatial phase and spatial frequency are held constant but the grating is moved at velocity $\nu/K$. A simple cell response to a drifting grating is highly modulated at frequency $\nu$, while a complex cell response to a drifting grating is elevated but relatively unmodulated. The differences between complex and simple cell responses are a direct consequence of the complex cell spatial phase insensitivity.

Previous models of complex cells generate spatial-phase invariant responses through converging sets of feedforward inputs with a wide range of spatial phase preferences but similar orientation and spatial frequency selectivities (Hubel and Wiesel, 1962; Mel et al., 1998). These models do not incorporate recurrent connections between complex cells, which are known to be particularly strong (Toyama et al., 1981). We propose that the spatial phase invariance of complex cell responses can arise from a broadening of spatial phase tuning by cortical amplification (Chance et al., 1998). The model neurons exhibit simple cell behavior when weakly coupled and complex cell behavior when strongly coupled, suggesting that the two classes of neurons in the primary visual cortex may arise from the same basic cortical circuit.

## 2 THE MODEL

The activity of neuron $i$ in the model network is characterized by a firing rate $r_i$. Each neuron sums feedforward and recurrent input and responds as described by the standard rate-model equation

$$\tau_r \frac{dr_i}{dt} = I_i + \sum W_{ij}r_j - r_i.$$

$I_i$ represents the feedforward input to cell $i$, $W_{ij}$ is the weight of the synapse from neuron $j$ to neuron $i$, and $\tau_r$ is a time constant. Previous studies have suggested that, for a neuron receiving many inputs, $\tau_r$ is small, closer to a synaptic time constant than the membrane time constant (Ben-Yishai et al., 1995; Treves, 1993). Thus we choose $\tau_r = 1$ ms.

The feedforward input describes the response of a simple cell with a Gabor receptive field

$$I_i = \left[ \int dx G_i(x) \int_0^\infty dt' H(t') s(x, t - t') \right]_+ ,$$

where $s(x,t)$ represents the contrast function of the visual stimulus and the notation $[\ ]_+$ indicates rectification. The temporal response function is (Adelson and Bergen, 1985)

$$H(t') = \exp(-\alpha t') \left( \frac{(\alpha t')^5}{5!} - \frac{(\alpha t')^7}{7!} \right),$$

where we use $\alpha = 1/$ms. The spatial filter is a Gabor function,

$$G = \exp\left( -\frac{x^2}{2\sigma_i^2} \right) \cos(k_i x - \phi_i),$$

where $\sigma_i$ determines the spatial extent of the receptive field, $k_i$ is the preferred spatial frequency, and $\phi_i$ is the preferred spatial phase. The values of $\phi_i$ are equally distributed

over the interval $[-180°, 180°)$. To give the neurons a realistic bandwidth, $\sigma_i$ is chosen such that $k_i\sigma_i = 2.5$. Initially we consider a simplified case in which $k_i = 1$ for all cells. Later we consider the spatial frequency selectivity of neurons in the network and allow the value of $k_i$ to range from 0 to 3.5 cycles/deg.

In this paper we assume that the model network describes one orientation column of the primary visual cortex, and thus all neurons have the same orientation tuning. All stimuli are of the optimal orientation for the network.

Spatial phase tuning is selectively broadened in the model because the strength of a recurrent connection between two neurons is independent of the spatial phase selectivities of their feedforward inputs. In the model with all $k_i = 1$, the recurrent input is determined by

$$W_{ij} = \frac{g}{(N-1)},$$

for all $i \neq j$. $N$ is the number of cells in the network, and $0 \leq g < g_{max}$, where $g_{max}$ is the largest value of $g$ for which the network remains stable. In this case $g_{max} = 1$.

## 3   RESULTS

The steady-state solution of the rate-model equation is given by $r_i = I_i + \sum W_{ij}r_j$. To solve this equation, we express the rates and feedfoward inputs in terms of a complete set of eigenvectors $\xi_i^\mu$ of the recurrent weight matrix, $\sum W_{ij}\xi_j^\mu = \lambda_\mu\xi_i^\mu$ for $\mu = 1, 2, \ldots, N$, where $\lambda_\mu$ are the eigenvalues. The solution is then

$$r_i = \sum_{\mu=1}^{N} \left( \frac{\xi_i^\mu}{1 - \lambda_\mu} \sum_{j=1}^{N} I_j\xi_j^\mu \right).$$

This equation displays the phenomenon of cortical amplification if one or more of the eigenvalues is near one. If we assume only one eigenvalue, $\lambda_1$, is close to one, the factor $1 - \lambda_1$ in the denominator causes the $\mu = 1$ term to dominate and we find $r_i \approx \xi_i^1 \sum I_j\xi_j^1(1 - \lambda_1)^{-1}$. The input combination $\sum I_j\xi_j^1$ dominates the response, determining selectivity, and this mode is amplified by a factor $1/(1 - \lambda_1)$. We refer to this amplification factor as the cortical gain.

In the case where $W_{ij} = g/(N-1)$ for $i \neq j$, the largest eigenvalue is $\lambda_1 = g$ and the corresponding eigenvector has all components equal to each other. For $g$ near one, the recurrent input to neuron $i$ is then proportional to $\sum_j[\cos(\Phi - \phi_j)]_+$ which, for large numbers of cells with uniformly placed preferred spatial phases $\phi_i$, is approximately independent of $\Phi$, the spatial phase of the stimulus. When $g$ is near zero, the network is at low gain and the response of neuron $i$ is roughly proportional to its feedforward input, $[\cos(\Phi - \phi_j)]_+$, and is sensitive to spatial phase.

The response properties of simple and complex cells to drifting and counterphase gratings are duplicated by the model neuron, as shown in figure 1. For low gain (gain = 1, top panels of figures 1A and 1B), the neuron acts as a simple cell and its activity is modulated at the same frequency as the stimulus ($\omega$ for counterphase gratings and $\nu$ for drifting gratings). At high gain (gain = 20), the neuron responds like a complex cell, exhibiting frequency doubling in the response to a counterphase grating (bottom panel of Figure 1A) and an elevated DC response to a drifting grating (bottom panel, Figure 1B). Intermediate gain (gain = 5) produces intermediate behavior (middle panels).

The basis of this model is that the amplified mode is independent of spatial phase. If the amplified mode depends on spatial frequency or orientation, neurons at high gain can be selective for these attributes. To show that the model can retain selectivity for other

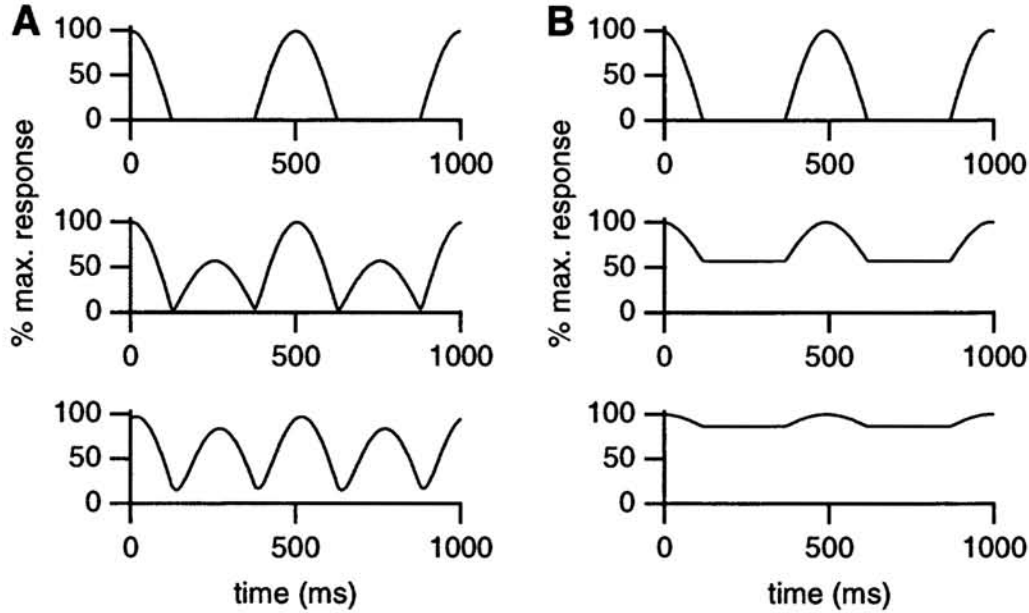

Figure 1: The effects of recurrent input on the responses of a neuron in the model network. The responses of one neuron to a 2 Hz counterphase grating (A) and to a 2 Hz drifting grating (B) are shown for different levels of network gain. From top to bottom in A and B, the gain of the network is one, five, and twenty.

stimulus characteristics while maintaining spatial phase insensitivity, we allowed the spatial frequency selectivity which each neuron receives from its feedforward input, $k_i$, to vary from neuron to neuron and also modified the recurrent weight matrix so that the strength of the connection between two neurons, $i$ and $j$, depends on $k_i - k_j$. The dependence is modeled as a difference of Gaussians, so the recurrent weight matrix is now

$$W_{ij} = \frac{g}{(N-1)} \left[ 2 \exp\left( -\frac{(k_i - k_j)^2}{2\sigma_c^2} \right) - \exp\left( -\frac{(k_i - k_j)^2}{2\sigma_s^2} \right) \right].$$

Thus neurons that receive feedforward input tuned for similar spatial frequencies excite each other and neurons that receive very differently tuned feedforward input inhibit each other. This produces complex cells that are tuned to a variety of spatial frequencies, but are still insensitive to spatial phase (see figure 2). The spatial frequency tuning curve width is primarily determined by $\sigma_c = 0.5$ cycle/deg and $\sigma_s = 1$ cycle/deg.

Cells within the same network do not have to exhibit the same level of gain. In previous figures, the gain of the network was determined by a parameter $g$ that described the strength of all the connections between neurons. In figure 3, the recurrent input to cell $i$ is determined by $W_{ij} = g_i/(N-1)$, where the values of $g_i$ are chosen randomly within the allowed range. The gain of each neuron depends on the value of $g_i$ for that neuron. As shown in figure 3, a range of complex and simple cell behaviors now coexist within the same network.

## 4   DISCUSSION

In the recurrent model we have presented, as in Hubel and Wiesel's feedforward model, the feedforward input to a complex cell arises from simple cells. Measurements by Alonso and

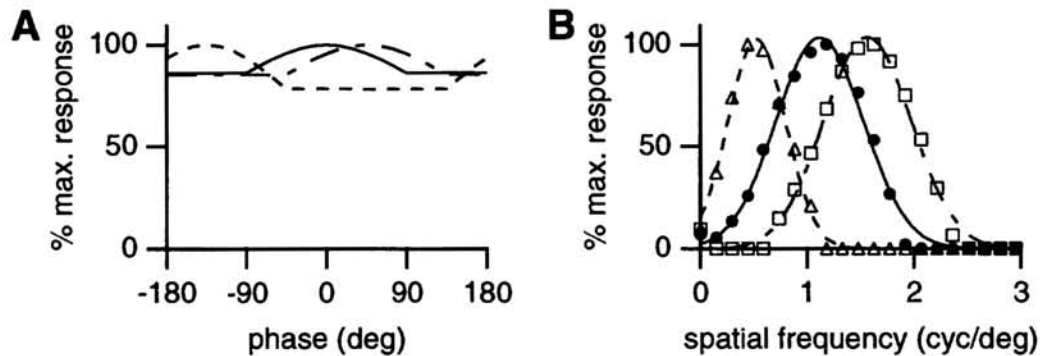

Figure 2: Neurons in a high-gain network can be selective for spatial frequency while remaining insensitive to spatial phase. Both spatial phase and spatial frequency tuning are included in the feedforward input. A) The spatial phase tuning curves of three representative neurons from a high-gain network. B) The spatial frequency tuning curves of the same three neurons as in A.

Martinez (1998) support this circuitry. However, direct excitatory input to complex cells arising from the LGN has also been reported (Hoffman and Stone, 1971; Singer et al., 1975; Ferster and Lindström, 1983). Supporting these measurements is evidence that certain stimuli can excite complex cells without strong excitation of simple cells (Hammond and Mackay, 1975, 1977; Movshon, 1975) and also that complex cells still respond when simple cells are silenced (Malpeli, 1983; Malpeli et al, 1986; Mignard and Malpeli, 1991). In accordance with this, the weak feedforward simple cell input in the recurrent model could probably be replaced by direct LGN input, as in the feedforward model of Mel et al. (1998).

The proposed model makes definite predictions about complex cell responses. If the phase-invariance of complex cell responses is due to recurrent interactions, manipulations that modify the balance between feedforward and recurrent drive should change the nature of the responses in a predictable manner. The model predicts that blocking local excitatory connections should turn complex cells into simple cells. Conversely, manipulations that increase cortical gain should make simple cells act more like complex cells. One way to increase cortical gain may be to block or partially block inhibition since this increases the influence of excitatory recurrent connections. Experiments along these lines have been performed, and blockade of inhibition does indeed cause simple cells to take on complex cell properties (Sillito, 1975; Shulz et al., 1993).

In a previous study, Hawken, Shapley, and Grosof (1996) noted that the temporal frequency tuning curves for complex cells are narrower for counterphase stimuli than for drifting stimuli. The recurrent model reproduces this result as long as the integration of synaptic inputs depends on temporal frequency. Such a dependence is provided, for example, by short-term synaptic depression (Chance et al., 1998). Hubel and Wiesel's feedforward model (1962) does not reproduce this effect, even with synaptic depression at the synapses.

We have presented a model of primary visual cortex in which complex cell response characteristics arise from recurrent amplification of simple cell responses. The complex cell responses in the high gain regime arise because recurrent connections selectively deamplify selectivity for spatial phase. Thus recurrent connections can act to generate invariant representation of input data. A similar mechanism could be used to produce responses that are independent of other stimulus attributes, such as size or orientation. Given the ubiquity of invariant representations in the visual pathway, this mechanism may have widespread use.

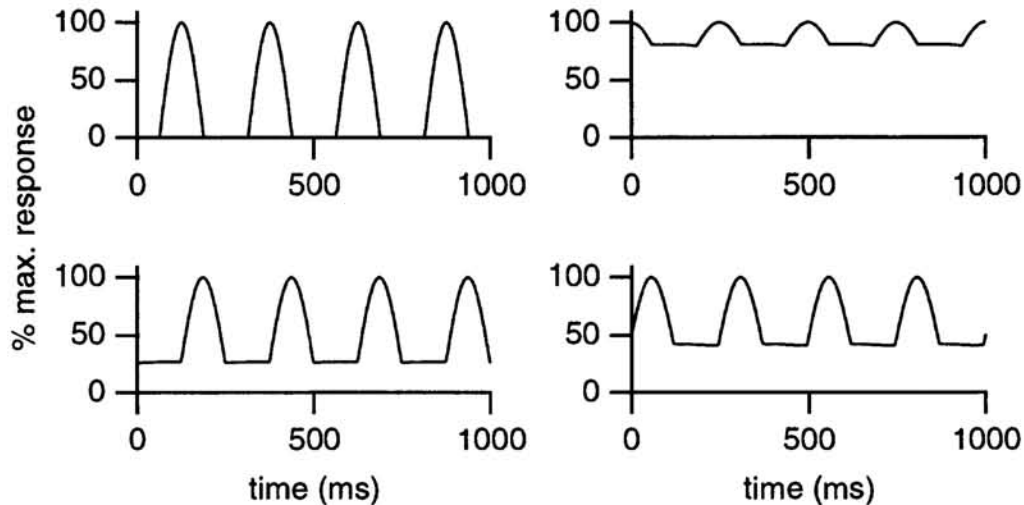

Figure 3: Responses to a 4 Hz drifting grating of four neurons from a large network consisting of a mixture of simple and complex cells. The two traces on the left represent simple cells and the two traces on the right represent complex cells.

## Acknowledgements

Research supported by the Sloan Center for Theoretical Neurobiology at Brandeis University, the National Science Foundation (DMS-95-03261), the W.M. Keck Foundation, the National Eye Institute (EY-11116), and the Alfred P. Sloan Foundation.

## References

Adelson, E. H. & Bergen, J. R. Spatiotemporal energy models for the perception of motion. *J. Opt. Soc. Am. A.* **2**, 284-299 (1985)

Alonso, J-M. & Martinez, L. M. Functional connectivity between simple cells and complex cells in cat striate cortex. *Nature Neuroscience* **1**, 395-403 (1998)

Ben-Yishai, R., Bar-Or, L. & Sompolinsky, H. Theory of orientation tuning in visual cortex. *Proc. Natl. Acad. Sci. USA* **92**, 3844-3848 (1995)

Chance F. S., Nelson S. B. & Abbott L. F. Complex cells as cortically amplified simple cells. *(submitted)*

Chung, S. & Ferster, D. Strength and orientation tuning of the thalamic input to simple cells revealed by electrically evoked cortical suppression. *Neuron* **20**, 1177-1189 (1998)

Douglas, R. J., Koch, C., Mahowald, M., Martin, K. A. C. & Suarez, H. H. Recurrent excitation in neocortical circuits. *Science* **269** 981-985 (1995)

Ferster, D., Chung, S. & Wheat, H. Orientation selectivity of thalamic input to simple cells of cat visual cortex. *Nature* **380**, 249-252 (1996)

Ferster, D. & Lindström, S. An intracellular analysis of geniculo-cortical connectivity in area 17 of the cat. *J. Physiol. (Lond)* **342**, 181-215 (1983)

Hammond, P. & MacKay, D. M. Differential responses of cat visual cortical cells to textured stimuli. *Exp. Brain Res.* **22**, 427-430 (1975)

Hammond, P. & MacKay, D. M. Differential responsiveness of simple and complex cells in cat striate cortex to visual texture. *Exp. Brain Res.* **30,** 275-296 (1977)

Hawken, M. J., Shapley, R. M. & Grosof, D. H. Temporal-frequency selectivity in monkey visual cortex. *Vis. Neurosci.* **13** 477-492 (1996)

Hoffman, K. P. & Stone, J. Conduction velocity of afferents of cat visual cortex: a correlation with cortical receptive field properties. *Brain Res.* **32,** 460-466 (1971)

Hubel, D. H. & Wiesel, T. N. Receptive fields, binocular interaction and functional architecture in the cat's visual cortex. *J. Physiol.* **160,** 106-154 (1962)

Malpeli, J. G. Activity of cells in area 17 of the cat in absence of input from layer A of lateral geniculate nucleus. *J. Neurophysiol.* **49,** 595-610 (1983)

Malpeli, J. G., Lee, C., Schwark, H. D. & Weyand, T. G. Cat area 17. I. Pattern of thalamic control of cortical layers. *J. Neurophysiol.* **56,** 1062-1073 (1986)

Mel, B. W., Ruderman, D. L. & Archie, K. A. Translation-invariant orientation tuning in visual complex cells could derive from intradendritic computations. *J. Neurosci.* **18** 4325-4334 (1998)

Mignard, M. & Malpeli, J. G. Paths of information flow through visual cortex. *Science* **251,** 1249-1251 (1991)

Movshon, J. A. The velocity tuning of single units in cat striate cortex. *J. Physiol.* **249,** 445-468 (1975)

Movshon, J., Thompson, I. & Tolhurst, D. Spatial summation in the receptive fields of simple cells in the cat's striate cortex. *J. Physiol. (Lond)* **283,** 53-77 (1978)

Movshon, J., Thompson, I. & Tolhurst, D. Receptive field organization of complex cells in cat's striate cortex. *J. Physiol. (Lond)* **283,** 79-99 (1978)

Shulz, D. E., Bringuier, B. & Frégnac, Y. A complex-like structure of simple visual cortical receptive fields is masked by GABA-A intracortical inhibition. *Soc. for Neurosci. Abs.* **19,** 628 (1993)

Sillito, A. M. The contribution of inhibitory mechanisms to the receptive field properties of neurones in the striate cortex of the cat. *J. Physiol. (Lond)* **250,** 305-329 (1975)

Singer, W., Tretter, F. & Cynader, M. Organization of cat striate cortex: a correlation of receptive-field properties with afferent and efferent connections. *J. Neurophysiol.* **38,** 1080-1098 (1975)

Somers, D. C., Nelson, S. B. & Sur, M. An emergent model of orientation selectivity in cat visual cortical simple cells. *J. Neurosci.* **15,** 5448-5465 (1995)

Sompolinsky, H. & Shapley, R. New perspectives on the mechanisms for orientation selectivity. *Current Opinion in Neurobiology* **7,** 514-522 (1997)

Toyama, K., Kimura, M. & Tanaka, K. Organization of cat visual cortex as investigated by cross-correlation technique. *J. Neurophysiol.* **46,** 202-214 (1981)

Treves, A. Mean-field analysis of neuronal spike dynamics. *Network* **4,** 259-284 (1993)